# Zero-Shot Event-Intensity Asymmetric Stereo via Visual Prompting from Image Domain

**Hanyue Lou**[#1,2]   **Jinxiu Liang**[#1,2]   **Minggui Teng**[1,2]   **Bin Fan**[3]   **Yong Xu**[4]   **Boxin Shi**[*1,2]

[1] State Key Laboratory of Multimedia Information Processing, School of Computer Science, Peking University
[2] National Engineering Research Center of Visual Technology, School of Computer Science, Peking University
[3] National Key Laboratory of General AI, School of Intelligence Science and Technology, Peking University
[4] School of Computer Science and Engineering, South China University of Technology
`{hylz,cssherryliang,minggui_teng,binfan,shiboxin}@pku.edu.cn`   `yxu@scut.edu.cn`

## Abstract

Event-intensity asymmetric stereo systems have emerged as a promising approach for robust 3D perception in dynamic and challenging environments by integrating event cameras with frame-based sensors in different views. However, existing methods often suffer from overfitting and poor generalization due to limited dataset sizes and lack of scene diversity in the event domain. To address these issues, we propose a zero-shot framework that utilizes monocular depth estimation and stereo matching models pretrained on diverse image datasets. Our approach introduces a visual prompting technique to align the representations of frames and events, allowing the use of off-the-shelf stereo models without additional training. Furthermore, we introduce a monocular cue-guided disparity refinement module to improve robustness across static and dynamic regions by incorporating monocular depth information from foundation models. Extensive experiments on real-world datasets demonstrate the superior zero-shot evaluation performance and enhanced generalization ability of our method compared to existing approaches.

## 1 Introduction

Stereo matching has witnessed significant advancements in recent years, driven by deep learning techniques and the availability of extensive training datasets in the image domain [18, 16, 20]. These advancements have enabled widespread applications in various fields, including mapping [10], navigation [21], 3D reconstruction [14, 9], motion estimation [7, 29], and image restoration [42, 19, 38]. Additionally, the abundance of unlabeled data on the internet has recently fueled the progress of monocular depth estimation [37, 25].

Event cameras report per-pixel relative intensity *changes* asynchronously at high temporal resolutions within a wide dynamic range [12], providing complementary sensory information alongside conventional frame-based cameras that capture *absolute* intensity values synchronously. Event-intensity asymmetric stereo matching has emerged as a promising approach to achieve robust performance in challenging conditions such as ultra-wide dynamic range and fast-moving scenes that cannot be faithfully captured by conventional frame-based cameras alone, by leveraging the complementary strengths of event and frame cameras in different views [30, 17, 46, 40, 4]. Despite the potential benefits, existing event-intensity asymmetric stereo approaches often rely on supervised learning or fine-tuning, requiring large amounts of labeled training data. Unfortunately, the shorter history of event-based sensors in commercial markets poses a scarcity of large-scale datasets essential for

---

[#] Equal contribution. [*] Corresponding author.
[1] Code available: `https://github.com/HYLZ-2019/ZEST`

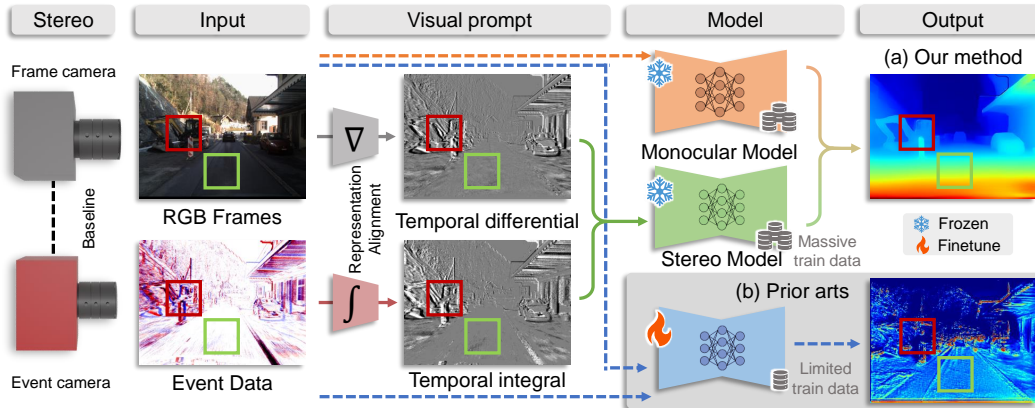

Figure 1: The proposed **Z**ero-shot **E**vent-intensity asymmetric **ST**ereo (ZEST) framework estimates disparity by finding correspondences between RGB frames and event data. (a) Our method conducts stereo matching by utilizing off-the-shelf stereo matching and monocular depth estimation models with frozen weights, and feeding them visual prompts tailored to the physical formulation of frames and events (temporal difference of frames and temporal integral of events, respectively). (b) In contrast, existing methods (*e.g.*, [40]) that rely on training data with known ground truth disparities often suffer from limited annotated data availability, thus leading to unsatisfactory results.

effective training and generalization. The scarcity of large-scale datasets in the event domain has resulted in overfitting and poor generalization to new environments or unseen disparity ranges [6].

Stereo models estimate disparity by establishing feature similarities between views, assuming that the two inputs are aligned in feature representation space. As events and frames capture relative differences and absolute values of intensity, respectively, they inherently possess a strong physical connection. This connection can be leveraged to convert them into intermediate representations with comparable appearance patterns. In the context of event-intensity asymmetric stereo, where training data are significantly limited compared to images, it is crucial and beneficial to develop a zero-shot approach that does not necessitate training data for modifying the underlying architecture or weights of the models. Considering the recent progress in image-based stereo matching [18, 16], where models trained on extensive datasets have exhibited effective zero-shot generalization, as well as the emerging techniques of "visual prompting" [32, 36, 1], which aims to adapt off-the-shelf models to new domains or modalities without modifying the model architecture or weights, we are motivated to utilize off-the-shelf models from the image domain with only modified inputs, rather than altering the weights, which requires substantial data.

Yet, several challenges impede the introduction of off-the-shelf models from the image domain to event in a zero-shot manner: 1) Significant modality gaps exist between events and frames (the red boxes in Figure 1), where events are triggered by temporal differences between frames exceeding predefined thresholds, compounded by sensor imperfections and stochastic electric noise. 2) In static regions where events cannot be triggered (the green boxes in Figure 1), no correspondences can be established, necessitating hallucination from the monocular model processing frames. However, these models typically provide relative disparities, whose distances from the actual metric are mostly calculated up to a global scale and bias.

In this paper, we propose a **Z**ero-shot **E**vent-intensity asymmetric **ST**ereo (ZEST) framework that leverages both monocular depth estimation and stereo matching models from the image domain, which is shown in Figure 1. To address the appearance gap between frames and events, we introduce a representation alignment module that considers the physical formulation from frames to events. The disparity map is then estimated from frames and events in different views using an off-the-shelf stereo model in the image domain. We further propose a monocular cue-guided disparity refinement module that re-renders these disparities by rescaling the relative depths predicted by a monocular depth estimation foundation model, enhancing robustness in regions with few events or textures. Our framework demonstrates superior performance among training-free methods for intensity-event asymmetric stereo matching and enhanced generalization across diverse real-world scenes. The

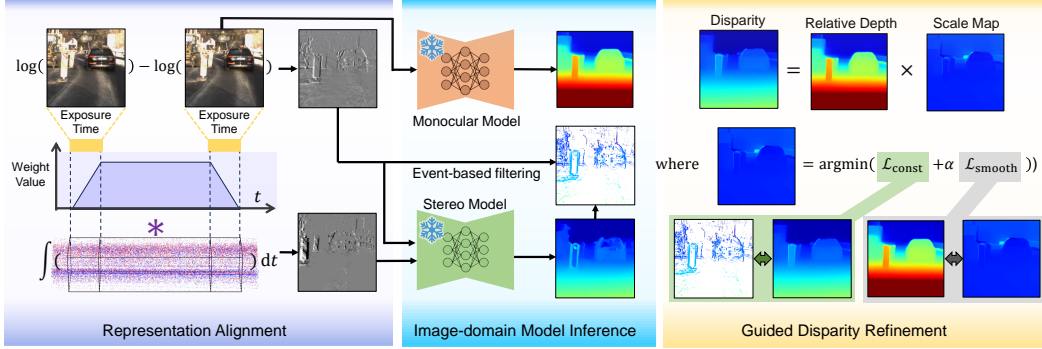

Figure 2: Overview of the proposed ZEST framework. The representation alignment module aligns frames and events, considering exposure time and event properties. This enables using an off-the-shelf stereo model to find correspondences. Disparity refinement then improves the estimates by minimizing differences between monocular depth prediction rescaled by an optimized scale map and binocular depth predictions, guided by event density confidence.

flexibility of our approach allows for seamless upgrades of the stereo and monocular models alongside advances in the related fields. Our main contributions are as follows:

- We present the first zero-shot event-intensity asymmetric stereo matching method that leverages off-the-shelf depth estimation models from the image domain.
- We introduce a visual prompting method for representation alignment between events and frames, enabling the utilization of off-the-shelf stereo models without modification.
- We propose a monocular cue-guided disparity refinement method for robustness in regions with few events or textures, inspired by recent advancements in monocular depth estimation.

## 2  Related Work

**Intensity-based stereo and monocular depth estimation.**  With the development of deep learning technology, significant progress has been made in stereo matching, with methods categorized based on their cost construction and aggregation approaches. Correlation-based methods [27, 35, 39, 8] and those using 3D convolutions [3, 5, 33] have achieved impressive performance. Recently, iterative optimization-based networks [20, 34, 41] have demonstrated superior accuracy and robustness. In monocular depth estimation, models like Depth Anything [37] and MiDaS [25] leverage extensive unlabeled data to estimate relative depth, enabling generalization across domains at the cost of unknown scale and shift.

**Event-based symmetric stereo.**  Event cameras capture pixel-level brightness changes asynchronously, offering advantages over conventional frame-based cameras. Event-based stereo depth estimation has emerged rapidly. Representative works include utilizing camera velocity [44] or estimating depth without explicit event matching [43]. Deep learning solutions have considered novel sequence embedding [28] and fusion of frame and event data [22, 23] for improved depth estimates in challenging scenarios. Recent efforts explore integrating off-the-shelf models from the image domain [6] to improve stereo matching performance by leveraging the inherent connection between frame and event data.

**Event-intensity asymmetric stereo.**  Event-frame asymmetric stereo matching leverages the complementary strengths of event and frame cameras. Traditional methods focused on aligning and fusing asynchronous event data with synchronous frame data using hand-crafted features [17] and traditional stereo matching algorithms [30]. Deep learning approaches [46, 40, 4] have been employed to learn complex mappings for event-frame fusion and dense depth estimation. However, these methods often suffer from overfitting and poor generalization due to limited dataset sizes and scene diversity in the event domain. Our work proposes a zero-shot approach that leverages disparity estimation models from the image domain by visual prompting, eliminating the need for additional training and improving generalization.

## 3 Method

**Overview** The proposed method aims to estimate depth from a frame-based camera and an event camera in different views, separated by a baseline distance. Without loss of generality, we assume that the frames are in the left view and the events are in the right view. Given consecutive rectified event-intensity pairs $(I^{\mathrm{L}}(\tau_i), E^{\mathrm{R}}(\tau_i))$ and $(I^{\mathrm{L}}(\tau_{i+1}), E^{\mathrm{R}}(\tau_{i+1}))$, our goal is to infer the corresponding disparity map $D(\tau_i)$ at timestamp $\tau_i$.

The overall framework of the proposed ZEST for event-intensity asymmetric stereo is shown in Figure 2, consisting of two components: the representation alignment module for aligning the frames in the left view and events in the right view into an intermediate representation space (Sec. 3.1), and the disparity refinement module for improving stereo matching results under the guidance of monocular model predictions (Sec. 3.2).

### 3.1 Event-intensity representation alignment for stereo matching

Stereo matching estimates depth by triangulation using pixel space representations, where stereo correspondence is established by finding similar patterns on a pixel-wise basis. With the advances in deep learning, modern stereo matching models are trained on massive data to estimate disparity. Due to the amount of training data and the diversity of real-world scenes, off-the-shelf models with frozen weights maintain robustness to different representations ranging from absolute values to relative changes in intensity, as shown in Figure 3. However, directly using these representations may not be optimal for event-intensity asymmetric stereo. This is because the event and frame data have fundamentally different characteristics, and a carefully designed intermediate representation can better bridge the appearance gap between them.

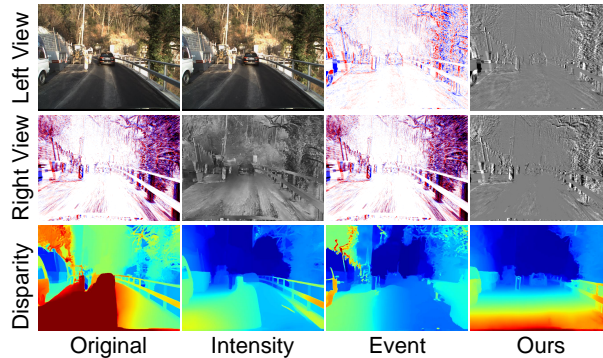

Figure 3: Visual comparisons of the disparity predicted by a stereo model [16] fed with inputs in the first two rows, which are aligned in the space of raw data, intensity (via [26]), events (via [15]), and intermediate (via the proposed method), respectively.

Inspired by this, we design an intermediate representation as a "visual prompt" to align the modalities in two views, enabling off-the-shelf stereo matching models to work for event-intensity asymmetric stereo. We will detail the formulation of the proposed intermediate representation in the following.

An event $e = (t, \boldsymbol{p}, \sigma)$ at the pixel $\boldsymbol{p} = (\boldsymbol{p}_x, \boldsymbol{p}_y)^\top$ and time $t$ is triggered whenever the logarithmic change of irradiance $I$ exceeds a pre-defined threshold $c$ $(> 0)$, *i.e.*,

$$\|\log I_{\boldsymbol{p}}(t) - \log I_{\boldsymbol{p}}(t - \Delta t)\| \geq c, \tag{1}$$

where $I(t)$ denotes the instantaneous intensity at time $t$, and the polarity $\sigma \in \{-1, +1\}$ indicates {negative, positive} brightness changes. We define $e_{\boldsymbol{p}}(t)$ as a function of continuous time $t$ such that,

$$e_{\boldsymbol{p}}(t) = \sigma \delta_\tau(t), \tag{2}$$

whenever there is an event $e = (\tau, \boldsymbol{p}, \sigma)$. Here, $\delta_\tau(t)$ is an impulse function, with unit integral, at time $\tau$, and the sequence of events is turned into a continuous-time signal, consisting of a sequence of impulses. There is such a function $e_{\boldsymbol{p}}(t)$ for each position $\boldsymbol{p}$ in the image. Since each pixel can be treated separately, we omit the subscripts $\boldsymbol{p}$. Given a reference timestamp $\tau_i$, assuming that there are latent sharp image sequences $I(\tau)$ with infinitesimal exposure time, their relationship between the corresponding events can be expressed as

$$I(\tau_{i+1}) = I(\tau_i) \exp\left(c \int_{\tau_i}^{\tau_{i+1}} e(t)dt\right). \tag{3}$$

Define the logarithmic brightness increment $\Delta L_i(t)$ from two consecutive frames as

$$\Delta L_i(t) = \log I(\tau_{i+1}) - \log I(\tau_i). \tag{4}$$

It can be approximated by events triggered during these frames as

$$\Delta \widehat{L}_i(t) = c \int_{\tau_i}^{\tau_{i+1}} e(t)dt. \tag{5}$$

In Eq. (5), the left-hand side represents the *temporal difference of frames*, while the right-hand side denotes the *temporal integral of events*. This formulation establishes an explicit intermediate representation that bridges the gap between frames and events with similar appearance, enabling correspondences to be found for stereo matching.

Now we turn to the frames captured in the real world, which have a non-negligible exposure time $2T$. A frame $I_{\tau,T}(t)$ with exposure time $[\tau - T, \tau + T]$ can be represented as the average of the latent image $I(t)$ over the exposure duration given a latent frame with a timestamp $\tau_0$ as reference [24], which can be formulated as

$$I_{\tau,T}(t) = \frac{1}{2T}I(\tau_0)\int_{\tau-T}^{\tau+T} \exp\left(c\int_{\tau_0}^{t} e(s)ds\right)dt. \tag{6}$$

Then, the difference between two consecutive logarithmic frames $L_{\tau_i}, L_{\tau_{i+1}}$ with exposure time $2T$ with the middle latent frame $I_{\tau_0}$ as reference can be formulated as

$$\Delta \widehat{L}_i(t) = \log\left(\int_{\tau_i-T}^{\tau_i+T} \exp\left(c\int_{\tau_0}^{t} e(s)ds\right)dt\right) - \log\left(\int_{\tau_{i+1}-T}^{\tau_{i+1}+T} \exp\left(c\int_{\tau_0}^{t} e(s)ds)dt\right)\right). \tag{7}$$

We use the temporal difference map $\Delta L(t)$ defined by consecutive frames in Eq. (4) and its approximation version defined from the temporal integral of events in Eq. (7) as explicit intermediate representations, respectively. The event trigger threshold $c$ is often unknown in real scenarios. However, Eq. (7) still holds after we normalize both sides of the equation for eliminating the unknown $c$, where percentile normalization is used for robustness. In practice, the calculations are done in discrete form, whose details can be found in the appendix.

Specifically, the disparity $D^{\text{bino}}$ at timestamp $t$ is estimated by stereo matching model $\mathcal{F}^{\text{bino}}$ as

$$D^{\text{bino}}(t) = \mathcal{F}^{\text{bino}}(\Delta L^{\text{L}}(t), \Delta \widehat{L}^{\text{R}}(t)). \tag{8}$$

As shown in Figure 3, the proposed event-intensity alignment method successfully finds appropriate visual prompts for the stereo models from the image domain, which helps to establish correspondences between the frames and events.

## 3.2 Monocular cue guided disparity refinement

In the context of event-intensity asymmetric stereo, stereo matching often faces challenges in establishing reliable correspondences, particularly in textureless regions of left images and static regions with sparse events in the right view. In contrast, monocular depth estimation directly infers depth maps from single images by leveraging monocular cues such as texture variations, gradients, occlusion, known object sizes, haze, and defocus. Off-the-shelf monocular depth estimation models, such as Depth Anything [37] and MiDaS [25], have demonstrated impressive "zero-shot cross-dataset transfer" capabilities, thanks to the relaxed requirements for training data in unsupervised learning.

Inspired by this, we propose a monocular cue-guided disparity refinement approach. However, there may be unknown scale and shift discrepancies between the predictions of the stereo and monocular models, which may vary spatially due to the absence of physically measurable information during monocular depth estimation. To address these factors, we model the desired refined disparity map as a locally linear transformation of the estimation from the monocular cue. Let $D^{\text{mono}}$ represent the disparity map predicted by a monocular depth estimation model $\mathcal{F}^{\text{mono}}$ from frame $I$, *i.e.*,

$$D^{\text{mono}} = \mathcal{F}^{\text{mono}}(I), \tag{9}$$

whose relationship with the binocular estimation $D^{\text{bino}}$ is assumed a linear transform as

$$D^{\text{bino}} \approx W \odot (D^{\text{mono}} + B), \tag{10}$$

where $\odot$ denotes the element-wise multiplication operation, and $W$ and $B$ denote the scale map and the shift map, respectively.

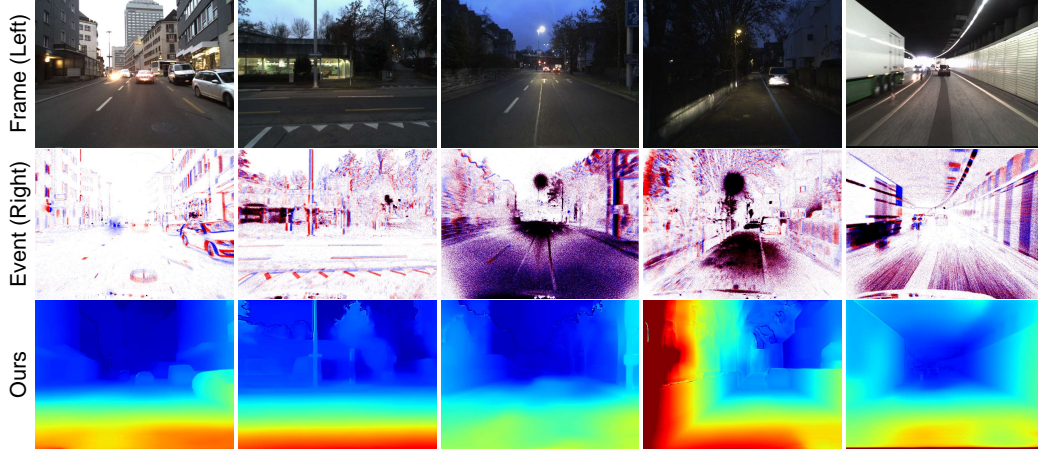

Figure 4: From left to right, our model exhibits impressive generalization abilities across a broad spectrum of varied scenes, encompassing sparse event scenes, richly textured environments, dimly lit settings, close-range captures, and high dynamic range situations.

To estimate the scale map $W$ and shift map $B$, we minimize the following loss function:

$$W^*, B^* = \text{argmin}_{W,B} \, \mathcal{L}_{\text{const}} + \alpha \mathcal{L}_{\text{smooth}}, \tag{11}$$

where the loss function involves several priors about the desired disparity map, and $\alpha$ is a regularization parameter to balance between them. We optimize this function using gradient descent with the Adam optimizer in PyTorch, running 500 iterations per image. Note that $D^{\text{bino}}$ is predicted by establishing correspondence between frames and events, which is more reliable where there are more events. Therefore, the temporal difference map $\Delta L^{\text{L}}(t)$ of frames is utilized to construct a confidence map $C$ to identify the density of events. Firstly, the estimated scale map $W$ and shift map $B$ should be consistent with the model defined in Eq. (10), which can be constrained by

$$\mathcal{L}_{\text{const}} = \sum_{\boldsymbol{p}} \left| C_{\boldsymbol{p}} (W_{\boldsymbol{p}}(D_{\boldsymbol{p}}^{\text{mono}} + B_{\boldsymbol{p}}) - D_{\boldsymbol{p}}^{\text{bino}}) \right|, \tag{12}$$

where the $\ell_1$ distance is utilized for its robustness to outliers. Secondly, the scales and biases for neighboring pixels should be similar, which can be derived by an edge-ware smoothness as

$$\mathcal{L}_{\text{smooth}} = \sum_{\boldsymbol{p}} \left( (|\partial_x W_{\boldsymbol{p}}| + |\partial_x B_{\boldsymbol{p}}|) e^{-\left(\partial_x D_{\boldsymbol{p}}^{\text{mono}}\right)^2} + (|\partial_y W_{\boldsymbol{p}}| + |\partial_y B_{\boldsymbol{p}}|) e^{-\left(\partial_y D_{\boldsymbol{p}}^{\text{mono}}\right)^2} \right). \tag{13}$$

This regularizer encourages local smoothness in the scale and shift maps. To ensure stability in the optimization steps for only one sample, a good initialization is necessary. While the shift map $B$ is simply initialized as all ones matrix $B^{(0)} = \mathbf{1}$, the scale map $W$ is initialized as

$$W^{(0)} = \frac{\sum_{\boldsymbol{p} \in \Omega_{\boldsymbol{p}}} \left( C_{\boldsymbol{p}} D_{\boldsymbol{p}}^{\text{bino}} / (D_{\boldsymbol{p}}^{\text{mono}} + B_{\boldsymbol{p}}^{(0)}) \right)}{\sum_{\boldsymbol{p} \in \Omega_{\boldsymbol{p}}} C_{\boldsymbol{p}}}, \tag{14}$$

where $\Omega_{\boldsymbol{p}}$ is a window centered at position $\boldsymbol{p}$. This loss term ensures the consistency modeled in Eq. (10) in regions with more events measured by $\Delta L$ in the beginning of the optimization. At the end of the optimization, the refined disparity map $\widehat{D}$ can be obtained by

$$\widehat{D} = W^* \odot (D^{\text{mono}} + B^*). \tag{15}$$

The proposed method effectively combines the strengths of both stereo matching and monocular depth estimation, leveraging the accurate but sparse disparity estimates from stereo matching to guide the refinement of the dense but relative depth estimates from monocular depth estimation.

Table 1: Quantitative comparisons of disparity estimation results with state-of-the-art methods from both event and image domains. The end-point-error (EPE), root mean square error (RMSE), 3-pixel error (3PE, %), and 2-pixel error (2PE, %) are adopted for evaluation. Zu, In, and Th denote the Zurich City, Interlaken, and Thun sequences on the DSEC [13] dataset, respectively. Red and orange highlights indicate the first and second best performing technique for each metric. ↑ (↓) indicates that higher (lower) values are better. The method with a gray background is the only one that does not adhere to the cross-dataset evaluation protocol.

| Method | EPE↓ | | | | RMSE↓ | | | | 3PE↓ | | | | 2PE↓ | | | |
|---|---|---|---|---|---|---|---|---|---|---|---|---|---|---|---|---|
| | Zu | In | Th | All | Zu | In | Th | All | Zu | In | Th | All | Zu | In | Th | All |
| SHEF [30] | 10.43 | 11.93 | 14.61 | 10.66 | 18.05 | 18.22 | 24.42 | 18.10 | 51.07 | 74.54 | 55.98 | 54.37 | 60.21 | 80.12 | 65.93 | 63.01 |
| HSM [17] | 8.65 | 8.34 | 8.42 | 8.60 | 19.11 | 17.96 | 19.16 | 18.95 | 32.55 | 36.40 | 30.87 | 33.08 | 42.10 | 45.77 | 38.15 | 42.60 |
| DAEI [40] | 12.43 | 12.09 | 13.89 | 12.39 | 15.66 | 15.44 | 17.12 | 15.63 | 87.10 | 86.02 | 89.97 | 86.96 | 91.48 | 90.74 | 93.58 | 91.39 |
| DAEI [40]† | - | 1.93 | - | - | - | 2.94 | - | - | - | 16.82 | - | - | - | 29.16 | - | - |
| Translate event into intensity on the right view | | | | | | | | | | | | | | | | |
| PSMNet-ETNet | 29.58 | 30.27 | 19.68 | 29.64 | 44.80 | 44.67 | 34.23 | 44.74 | 80.09 | 85.98 | 75.93 | 80.90 | 89.33 | 91.95 | 87.14 | 89.69 |
| CR-ETNet | 27.99 | 19.20 | 5.25 | 26.67 | 34.31 | 26.93 | 12.31 | 33.19 | 31.90 | 25.44 | 12.75 | 30.92 | 40.46 | 35.64 | 20.42 | 39.71 |
| DS-ETNet | 20.84 | 24.04 | 2.93 | 21.22 | 29.32 | 40.45 | 5.67 | 30.78 | 34.19 | 36.18 | 23.00 | 34.42 | 43.46 | 47.43 | 33.10 | 43.98 |
| PSMNet-E2VID | 29.50 | 26.69 | 25.07 | 29.09 | 44.84 | 38.74 | 39.42 | 43.96 | 81.43 | 84.43 | 82.31 | 81.86 | 90.15 | 91.33 | 90.53 | 90.32 |
| CR-E2VID | 24.65 | 7.70 | 3.67 | 22.20 | 30.60 | 12.17 | 8.15 | 27.94 | 27.78 | 12.75 | 9.51 | 25.60 | 35.06 | 21.23 | 15.18 | 33.05 |
| DS-E2VID | 13.30 | 17.40 | 2.37 | 13.83 | 20.02 | 30.72 | 4.28 | 21.46 | 28.20 | 28.83 | 20.23 | 28.25 | 36.38 | 38.70 | 29.09 | 36.68 |
| Translate intensity into event on the left view | | | | | | | | | | | | | | | | |
| CFF-v2e | 9.86 | 12.07 | 7.81 | 10.17 | 14.69 | 16.55 | 11.36 | 14.93 | 60.34 | 71.95 | 62.77 | 61.97 | 68.55 | 79.73 | 73.03 | 70.14 |
| ZEST: Translate into intermediate representation on both views (Ours) | | | | | | | | | | | | | | | | |
| Ours-CR-MiDaS | 3.64 | 8.79 | 2.21 | 4.35 | 4.60 | 9.68 | 3.23 | 5.30 | 28.68 | 33.07 | 21.61 | 29.26 | 48.02 | 51.52 | 41.12 | 48.48 |
| Ours-DS-MiDaS | 2.24 | 7.66 | 1.68 | 3.00 | 3.46 | 12.07 | 2.82 | 4.66 | 14.48 | 17.57 | 12.46 | 14.91 | 26.31 | 28.58 | 22.39 | 26.61 |
| Ours-CR-DA | 3.18 | 9.00 | 1.31 | 3.99 | 4.27 | 9.93 | 2.40 | 5.05 | 9.75 | 10.48 | 7.26 | 9.84 | 18.76 | 18.05 | 14.23 | 18.64 |
| Ours-DS-DA | 2.24 | 7.66 | 1.71 | 2.99 | 3.44 | 12.05 | 2.86 | 4.64 | 14.67 | 17.44 | 13.11 | 15.05 | 26.14 | 28.21 | 22.83 | 26.41 |

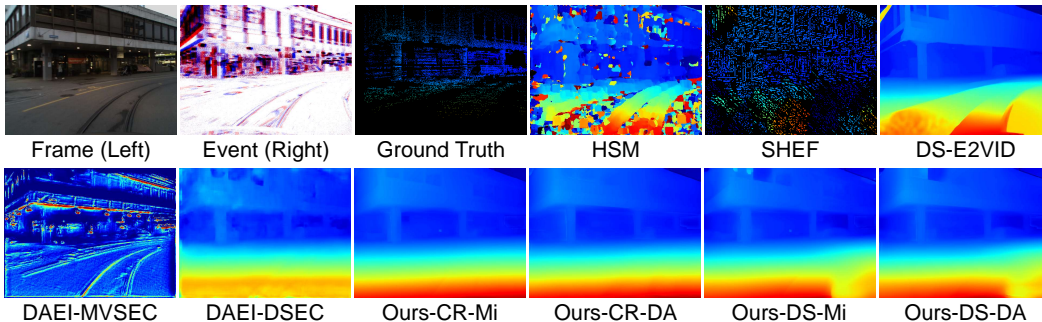

Frame (Left) | Event (Right) | Ground Truth | HSM | SHEF | DS-E2VID

DAEI-MVSEC | DAEI-DSEC | Ours-CR-Mi | Ours-CR-DA | Ours-DS-Mi | Ours-DS-DA

Figure 5: Visual quality comparison of disparity estimation results among state-of-the-art methods (HSM [17], SHEF [30], DAEI [40] trained on MVSEC [45] and DSEC [13], respectively) and the proposed ZEST with various stereo matching models (CR and DS) and monocular depth estimation models (Mi and DA). The baseline method with the best EPE and RMSE metrics, *i.e.*, DS-E2VID, is also included for comparison.

## 4 Experiments

**Dataset.** We evaluate the proposed ZEST framework on the widely-used benchmark dataset for event-intensity stereo matching, the DSEC dataset [13], a large-scale high-quality driving dataset with challenging scenes. It consists of synchronized event and frame streams captured from a stereo setup in a wider range of challenging scenarios, including fast motion, high dynamic range, and low light conditions. Specifically, it provides high-resolution ($640 \times 480$) stereo event streams captured in outdoor driving scenes using Prophesee Gen 3.1 event cameras across 53 outdoor driving scenarios under diverse lighting. Without specification, all 41 sequences (5 Interlaken sequences, 1 Thun sequence, and 35 Zurich City sequences) in the training set are adopted for evaluation, as the official "test" split lacks ground truth disparity. To address boundary issues with certain methods, such as E2VID's inability to reconstruct the initial frames without prior events, we exclude the first and last 10 frames of each sequence from metric calculations. To assess generalization, we also evaluate on the MVSEC [45] and M3ED [2] datasets. MVSEC [45], the pioneering stereo event dataset, includes ground truth depth maps in diverse scenarios with DAVIS346 event cameras ($346 \times 260$ resolution). We use three subsets for MVSEC evaluation: indoor_flying1 (500–1500), indoor_flying2 (500–2000),

Table 2: Quantitative results of the proposed zero-shot disparity estimation method on the MVSEC [45] dataset.

| Method | EPE↓ | | | | RMSE↓ | | | | 3PE↓ | | | | 2PE↓ | | | |
|---|---|---|---|---|---|---|---|---|---|---|---|---|---|---|---|---|
| | S1 | S2 | S3 | All | S1 | S2 | S3 | All | S1 | S2 | S3 | All | S1 | S2 | S3 | All |
| HSM [17] | 9.64 | 12.98 | 9.09 | 10.57 | 11.43 | 15.81 | 10.87 | 12.70 | 82.22 | 83.75 | 79.48 | 81.82 | 85.96 | 89.03 | 83.68 | 86.22 |
| DAEI [40] | 1.08 | - | - | - | 1.55 | - | - | - | 7.73 | - | - | - | 15.73 | - | - | - |
| CR-E2VID | 16.33 | 23.62 | 18.01 | 19.32 | 16.94 | 24.64 | 18.55 | 20.04 | 79.31 | 88.24 | 83.39 | 83.64 | 82.12 | 92.05 | 88.42 | 87.53 |
| Ours-CR-DA | 3.34 | 7.19 | 5.13 | 5.22 | 3.83 | 7.83 | 5.60 | 5.76 | 19.06 | 41.72 | 28.66 | 29.82 | 33.22 | 61.38 | 43.97 | 46.19 |

indoor_flying3 (500-2500), denoted as S1, S2, and S3, respectively. Given the differing frame and depth rates, we select the depth map closest to each image frame timestamp. The M3ED [2] dataset captures unique urban and forest scenes with Prophesee EVK4 HD event cameras (1280 × 720 resolution). We use the car_urban_day_horse (300-700) sequence for evaluation.

**Metrics** We use the standard evaluation metrics for stereo matching, including the mean absolute error (MAE), root mean squared error (RMSE), and the percentage of pixels with errors larger than a threshold (*e.g.*, 1, 2, or 3 pixels).

**Compared methods.** We compare the performance of the proposed ZEST framework with state-of-the-art event-intensity stereo matching methods, including both traditional and deep-learning-based approaches. For traditional methods, we consider SHEF [30] and HSM [17]. For deep-learning-based methods, we compare against a state-of-the-art method DAEI [40], originally trained on the MVSEC [45] dataset (S2 and S3 splits), which has limited generalizability to DSEC. We also test a variant of DAEI (denoted DAEI[†]), finetuned on the Zurich and Thun sequences in DSEC for 34 epochs and evaluated on both DSEC and M3ED.

**Baselines.** We also include several baseline methods that directly apply the off-the-shelf stereo models to the event and frame images without extra representation alignment or disparity refinement. To align the different modalities between the left and right views, we consider two cases, event-to-intensity and intensity-to-event, respectively. In the case of event-to-intensity, events in the right view are reconstructed into a gray image using E2VID [26] and ETNet [31] and paired with frames in the left view. The off-the-shelf image-based stereo models used include PSMNet (checkpoint trained on KITTI2015) [3], CREStereo (CR, checkpoint trained on ETH3D) [18], and DynamicStereo (DS, checkpoint trained on DynamicReplica and SceneFlow) [16]. In the case of intensity-to-event, consecutive frames in the left view are converted by v2e [15], which are then fed to the off-the-shelf event-based stereo models CFF [23] together with the events in the right view. As for the proposed ZEST framework, we adopt CR and DS for the stereo models, and Depth Anything (DA, checkpoint Depth-Anything-Large, 335.3M parameters) [37] and MiDaS (Mi, checkpoint BEiT-L-512, 345M parameters) [25] for monocular depth estimation. Throughout this paper, we use abbreviations to denote specific combinations of modality alignment, stereo models, and monocular models. For example, the combination of the proposed technique, CREStereo, and Depth Anything is referred to as "Ours-CR-DA". All results for comparison are produced from their official codes and models with recommended hyperparameters provided on public available sources or provided by the authors.

**Compute environment setup.** All models are tested on an Intel i7-13700K CPU and a single NVIDIA RTX 4090 GPU. While representation alignment and disparity refinement modules can run on either CPU or GPU, stereo and monocular depth estimation models require GPU acceleration.

### 4.1 Comparisons with prior arts

Quantitative results on the benchmark dataset DSEC [13] are reported in Table 1, demonstrating the proposed method's superior performance. The quantitative analysis revealed that our framework consistently outperformed almost all compared methods and baselines across every metric, except for DAEI[†] [40] and the baseline E2VID-CR. While all other methods

Table 3: Quantitative results of the proposed zero-shot disparity estimation method on the M3ED [2] dataset.

| Method | EPE↓ | RMSE↓ | 3PE↓ | 2PE↓ |
|---|---|---|---|---|
| HSM [17] | 12.39 | 14.27 | 90.87 | 92.58 |
| DAEI [40][†] | 20.07 | 22.19 | 93.12 | 95.47 |
| CR-E2VID | 2.10 | 4.02 | 17.69 | 25.45 |
| Ours-CR-DA | 2.06 | 3.39 | 19.04 | 29.02 |

are evaluated in a cross-dataset manner, DAEI[†] [40] is the only method that is evaluated in an in-dataset manner, which is trained and tested on the DSEC [13]. Therefore, it is not surprising that

Table 4: Quantitative results of ablation studies on the interlaken_00_c sequence of the DSEC [13] dataset. Compared to Table 1, 1-pixel error (1PE, %) is also utilized for evaluation.

| Setting | Rep. alignment | Monocular cue | EPE↓ | RMSE↓ | 3PE↓ | 2PE↓ | 1PE↓ |
|---|---|---|---|---|---|---|---|
| 1) DS w/ E2VID-right | ✗ | ✗ | 43.02 | 55.79 | 87.63 | 91.57 | 95.98 |
| 2) DS w/ v2e-left | ✗ | ✗ | 13.90 | 20.26 | 72.68 | 81.44 | 90.65 |
| 3) DS w/ spatial gradients | ✗ | ✗ | 19.01 | 23.76 | 78.88 | 86.24 | 93.05 |
| 4) DS w/ spatial gradients + DA_Large | ✗ | ✓ | 19.09 | 23.57 | 81.38 | 88.20 | 94.49 |
| 5) DA_Large | ✗ | ✓ | 35.85 | 41.34 | 99.05 | 99.35 | 99.65 |
| 6) DA_Large w/ GT scale | ✗ | ✓ | 2.40 | 3.16 | 28.35 | 47.45 | 72.07 |
| 7) Ours-DS w/o DA | ✓ | ✗ | 1.49 | 2.85 | 7.77 | 16.84 | 47.84 |
| 8) Ours-DS-DA_Large | ✓ | ✓ | 1.41 | 2.62 | 7.22 | 16.00 | 46.37 |
| 9) Ours-DS-DA_Base | ✓ | ✓ | 1.39 | 2.59 | 6.96 | 15.60 | 46.17 |
| 10) Ours-DS-DA_Small | ✓ | ✓ | 1.42 | 2.64 | 7.30 | 15.94 | 46.95 |

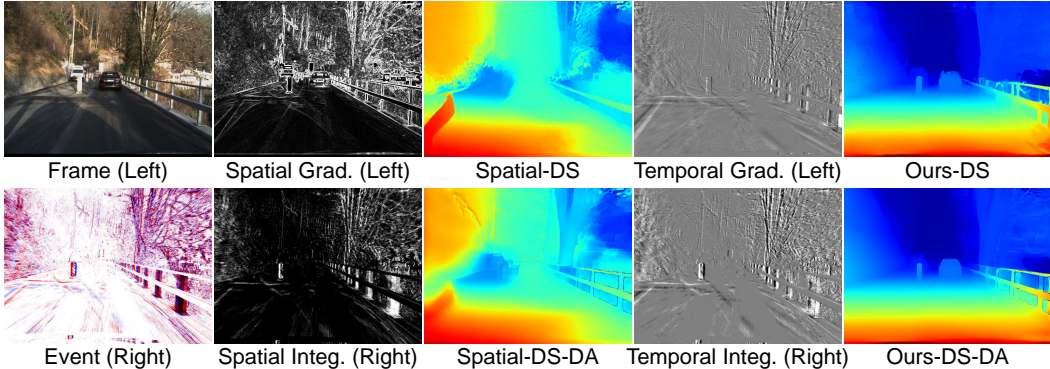

| Frame (Left) | Spatial Grad. (Left) | Spatial-DS | Temporal Grad. (Left) | Ours-DS |
| Event (Right) | Spatial Integ. (Right) | Spatial-DS-DA | Temporal Integ. (Right) | Ours-DS-DA |

Figure 6: Visual comparison of the disparity results of a stereo matching method DS using different representations and the proposed approach. From left to right: inputs, spatial gradients of frames and spatial integral of events (via [11]), their corresponding disparity result, the proposed representation, *i.e.*, the temporal difference of the frame and the temporal integral of the events, and their corresponding disparity result.

they achieve almost the best performance. Surprisingly, most of the variants of the proposed ZEST framework outperform this method in terms of 3PE metric, which demonstrates the effectiveness of the proposed method. The performance of the baseline CR-E2VID achieves good performance in terms of the 3PE and 2PE metrics in some sequences, although worse than the proposed method in all sequences. Quantitative results on MVSEC [45] and M3ED [2] datasets are shown in Tables 2 and 3, respectively, which demonstrate ZEST's robust generalization across diverse scenarios.

The visual results across varied scenes shown in Figure 4 demonstrate the generalizability of our method. Visual comparisons on DSEC are shown in Figure 5. For the baselines, we include DS-E2VID [16], which achieved the best performance in terms of EPE and RMSE metrics. They highlight the superior quality of our framework, generating depth maps with significantly enhanced sharpness, intricate details, and improved dynamic accuracy compared to the compared methods.

## 4.2 Ablation study

To validate the effectiveness of each component in the proposed ZEST framework and analyze their contributions to the overall performance, we conduct a series of ablation studies to evaluate the impact of the representation alignment module and the monocular cue-guided disparity refinement module.

**Impact of the representation alignment module.** To assess the importance of the representation alignment module, we compare the performance of ZEST with and without this module. Quantitative results are shown in Table 4. In the absence of the proposed representation alignment, we feed the off-the-shelf stereo matching model DS with: 1) original frames and frames generated from events in the right view via E2VID (Figure 6); 2) events generated from frames in the left view via v2e and events in the right view; 3) the spatial gradient of frames in the left view and the spatial integral of events in the right view using [11]; and 7) the proposed representation alignment module. Among these settings, the proposed module achieves the best performance. This highlights the effectiveness of our approach in bridging the modality gap between events and frames, enabling the successful

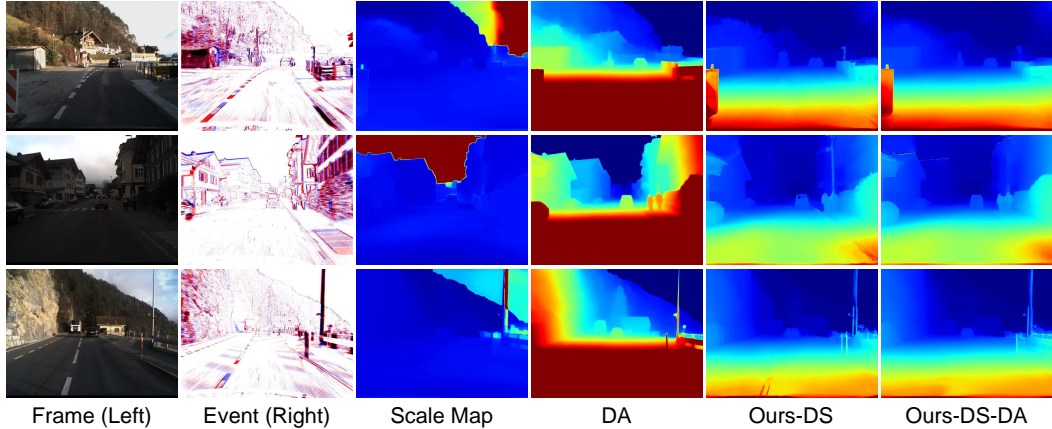

| Frame (Left) | Event (Right) | Scale Map | DA | Ours-DS | Ours-DS-DA |

Figure 7: Visual comparison of the effectiveness of the monocular cue-guided disparity refinement module. From left to right: input frames, input events, scale map results, disparity results from the monocular model DA alone, results from the proposed method without DA, and results with DA incorporated.

application of off-the-shelf stereo matching models. The corresponding qualitative results are shown in Figures 3 and 6, respectively.

**Impact of the disparity refinement module.** To validate the effectiveness of each component, we compare the proposed method with its five variants: 3) stereo matching model fed with the spatial gradient of frames in the left view and the spatial integral of events in the right view; 4) the results of 3) refined by a monocular depth estimation; 5) only the monocular depth estimation model DA; 6) the results of (5) rescaled by a global scale calculated from the ground truth disparity; 7) the proposed method without disparity refinement; and 8) the proposed method with DA for disparity refinement. The effectiveness of the introduction of monocular depth estimation can be shown by comparing 7) and 8) and the corresponding qualitative results are shown in Figure 7, whose results demonstrate more natural edges with DA. However, the disparity refinement module fails when the stereo matching results are totally not reliable, as shown in the comparison between 3) and 4). As shown in Figures 6 and 7, the disparity refinement module improves sharp depth boundaries for objects, such as cars, in challenging scenarios with sparse events or low-texture regions.

**Impact of monocular depth estimation model size for refinement.** Our framework's modular design allows for deployment with lighter-weight models, ideal for resource-limited environments. While we use the DA_Large (335.3M parameters) for results in Table 1, we also evaluated compact alternatives, DA_Base (97.5M) and DA_Small (24.8M), as shown in 9) and 10) in Table 4. These alternatives provide substantial speed gains with acceptable accuracy trade-offs.

## 5 Conclusion

We introduce ZEST, a novel zero-shot event-intensity stereo matching framework that utilizes cutting-edge image domain models for accurate disparity estimation without training data. ZEST addresses the modality gap and labeled data scarcity in the event domain through representation alignment and monocular cue-guided disparity refinement. Experiments on DSEC show ZEST outperforms state-of-the-art methods in cross-dataset evaluation. Ablation studies validate the effectiveness of each component, highlighting the importance of representation alignment, model integration versatility, and monocular cue-guided refinement benefits.

**Acknowledgments.** This work was supported by National Natural Science Foundation of China (Grant No. 62136001, 62302019, 62088102, 62472179), Beijing Natural Science Foundation (Grant No. L233024), Beijing Municipal Science & Technology Commission, Administrative Commission of Zhongguancun Science Park (Grant No. Z241100003524012), National Key Research and Development Program of China (Grant No. 2024YFE0105400). Bin Fan was also supported by China Postdoctoral Science Foundation (Grant No. 2024M750101) and China National Postdoctoral Program for Innovative Talents (Grant No. BX20230013).

# References

[1] Hyojin Bahng, Ali Jahanian, Swami Sankaranarayanan, and Phillip Isola. Exploring visual prompts for adapting large-scale models. *arXiv preprint arXiv:2203.17274*, 2022.

[2] Kenneth Chaney, Fernando Cladera, Ziyun Wang, Anthony Bisulco, M. Ani Hsieh, Christopher Korpela, Vijay Kumar, Camillo J. Taylor, and Kostas Daniilidis. M3ED: Multi-Robot, Multi-Sensor, Multi-Environment Event Dataset. In *Proc. of Computer Vision and Pattern Recognition Workshops*, 2023.

[3] Jia-Ren Chang and Yong-Sheng Chen. Pyramid stereo matching network. In *Proc. of Computer Vision and Pattern Recognition*, 2018.

[4] Xihao Chen, Wenming Weng, Yueyi Zhang, and Zhiwei Xiong. Depth from asymmetric frame-event stereo: A divide-and-conquer approach. In *Proc. of Winter Conference on Applications of Computer Vision*, 2024.

[5] Xinjing Cheng, Peng Wang, and Ruigang Yang. Learning depth with convolutional spatial propagation network. *IEEE Transactions on Pattern Analysis and Machine Intelligence*, 42(10): 2361–2379, 2020.

[6] Hoonhee Cho, Jegyeong Cho, and Kuk-Jin Yoon. Learning adaptive dense event stereo from the image domain. In *Proc. of Computer Vision and Pattern Recognition*, 2023.

[7] Bin Fan, Yuchao Dai, and Ke Wang. Rolling-shutter-stereo-aware motion estimation and image correction. *Computer Vision and Image Understanding*, 213:103296, 2021.

[8] Bin Fan, Ke Wang, Yuchao Dai, and Mingyi He. RS-DPSNet: Deep plane sweep network for rolling shutter stereo images. *IEEE Signal Processing Letters*, 28:1550–1554, 2021.

[9] Bin Fan, Yuchao Dai, Zhiyuan Zhang, and Ke Wang. Differential SfM and image correction for a rolling shutter stereo rig. *Image and Vision Computing*, 124:104492, 2022.

[10] Jorge Fuentes-Pacheco, José Ruiz-Ascencio, and Juan Manuel Rendón-Mancha. Visual simultaneous localization and mapping: A survey. *Artificial Intelligence Review*, 43(1):55–81, 2015.

[11] Guillermo Gallego, Henri Rebecq, and Davide Scaramuzza. A unifying contrast maximization framework for event cameras, with applications to motion, depth, and optical flow estimation. In *Proc. of Computer Vision and Pattern Recognition*, 2018.

[12] Guillermo Gallego, Tobi Delbruck, Garrick Michael Orchard, Chiara Bartolozzi, Brian Taba, Andrea Censi, Stefan Leutenegger, Andrew Davison, Jorg Conradt, Kostas Daniilidis, and Davide Scaramuzza. Event-based vision: A survey. *IEEE Transactions on Pattern Analysis and Machine Intelligence*, 44(1):154–180, 2020.

[13] Mathias Gehrig, Willem Aarents, Daniel Gehrig, and Davide Scaramuzza. DSEC: A stereo event camera dataset for driving scenarios. *IEEE Robotics and Automation Letters*, 6(3):4947–4954, 2021.

[14] Leonardo Gomes, Olga Regina Pereira Bellon, and Luciano Silva. 3D reconstruction methods for digital preservation of cultural heritage: A survey. *Pattern Recognition Letters*, 50:3–14, 2014.

[15] Yuhuang Hu, Shih-Chii Liu, and Tobi Delbruck. v2e: From video frames to realistic dvs events. In *Proc. of Computer Vision and Pattern Recognition Workshops*, 2021.

[16] Nikita Karaev, Ignacio Rocco, Benjamin Graham, Natalia Neverova, Andrea Vedaldi, and Christian Rupprecht. DynamicStereo: Consistent dynamic depth from stereo videos. In *Proc. of Computer Vision and Pattern Recognition*, 2023.

[17] Haram Kim, Sangil Lee, Junha Kim, and H. Jin Kim. Real-Time Hetero-Stereo Matching for Event and Frame Camera With Aligned Events Using Maximum Shift Distance. *IEEE Robotics and Automation Letters*, 8(1):416–423, 2023.

[18] Jiankun Li, Peisen Wang, Pengfei Xiong, Tao Cai, Ziwei Yan, Lei Yang, Jiangyu Liu, Haoqiang Fan, and Shuaicheng Liu. Practical stereo matching via cascaded recurrent network with adaptive correlation. In *Proc. of Computer Vision and Pattern Recognition*, 2022.

[19] Jinxiu Liang, Yixin Yang, Boyu Li, Peiqi Duan, Yong Xu, and Boxin Shi. Coherent event guided low-light video enhancement. In *Proc. of International Conference on Computer Vision*, 2023.

[20] Lahav Lipson, Zachary Teed, and Jia Deng. RAFT-Stereo: Multilevel recurrent field transforms for stereo matching. In *Proc. of International Conference on 3D Vision*, 2021.

[21] Daniel J. Mirota, Masaru Ishii, and Gregory D. Hager. Vision-based navigation in image-guided interventions. *Annual Review of Biomedical Engineering*, 13:297–319, 2011.

[22] Mohammad Mostafavi, Kuk-Jin Yoon, and Jonghyun Choi. Event-intensity stereo: Estimating depth by the best of both worlds. In *Proc. of International Conference on Computer Vision*, 2021.

[23] Yeongwoo Nam, Mohammad Mostafavi, Kuk-Jin Yoon, and Jonghyun Choi. Stereo depth from events cameras: Concentrate and focus on the future. In *Proc. of Computer Vision and Pattern Recognition*, 2022.

[24] Liyuan Pan, Richard Hartley, Cedric Scheerlinck, Miaomiao Liu, Xin Yu, and Yuchao Dai. High frame rate video reconstruction based on an event camera. *IEEE Transactions on Pattern Analysis and Machine Intelligence*, 44(5):2519–2533, 2020.

[25] René Ranftl, Katrin Lasinger, David Hafner, Konrad Schindler, and Vladlen Koltun. Towards robust monocular depth estimation: Mixing datasets for zero-shot cross-dataset transfer. *IEEE Transactions on Pattern Analysis and Machine Intelligence*, 44(3):1623–1637, 2022.

[26] Timo Stoffregen, Cedric Scheerlinck, Davide Scaramuzza, Tom Drummond, Nick Barnes, Lindsay Kleeman, and Robert Mahony. Reducing the sim-to-real gap for event cameras. In *Proc. of European Conference on Computer Vision*, 2020.

[27] Alessio Tonioni, Oscar Rahnama, Thomas Joy, Luigi Di Stefano, Thalaiyasingam Ajanthan, and Philip H. S. Torr. Learning to adapt for stereo. In *Proc. of Computer Vision and Pattern Recognition*, 2019.

[28] Stepan Tulyakov, Francois Fleuret, Martin Kiefel, Peter Gehler, and Michael Hirsch. Learning an event sequence embedding for dense event-based deep stereo. In *Proc. of International Conference on Computer Vision*, 2019.

[29] Ke Wang, Bin Fan, and Yuchao Dai. Relative pose estimation for stereo rolling shutter cameras. In *Proc. of International Conference on Image Processing*, 2020.

[30] Ziwei Wang, Liyuan Pan, Yonhon Ng, Zheyu Zhuang, and Robert Mahony. Stereo hybrid event-frame (SHEF) cameras for 3D perception. In *Proc. of International Conference on Intelligent Robots and Systems*, 2021.

[31] Wenming Weng, Yueyi Zhang, and Zhiwei Xiong. Event-based video reconstruction using Transformer. In *Proc. of International Conference on Computer Vision*, 2021.

[32] Chen Henry Wu, Saman Motamed, Shaunak Srivastava, and Fernando De la Torre. Generative visual prompt: Unifying distributional control of pre-trained generative models. In *Adv. of Neural Information Processing Systems*, 2022.

[33] Gangwei Xu, Junda Cheng, Peng Guo, and Xin Yang. Attention concatenation volume for accurate and efficient stereo matching. In *Proc. of Computer Vision and Pattern Recognition*, 2022.

[34] Gangwei Xu, Xianqi Wang, Xiaohuan Ding, and Xin Yang. Iterative geometry encoding volume for stereo matching. In *Proc. of Computer Vision and Pattern Recognition*, 2023.

[35] Haofei Xu and Juyong Zhang. AANet: Adaptive aggregation network for efficient stereo matching. In *Proc. of Computer Vision and Pattern Recognition*, 2020.

[36] Lingfeng Yang, Yueze Wang, Xiang Li, Xinlong Wang, and Jian Yang. Fine-grained visual prompting. In *Adv. of Neural Information Processing Systems*, 2023.

[37] Lihe Yang, Bingyi Kang, Zilong Huang, Xiaogang Xu, Jiashi Feng, and Hengshuang Zhao. Depth Anything: Unleashing the power of large-scale unlabeled data. In *Proc. of Computer Vision and Pattern Recognition*, 2024.

[38] Yixin Yang, Jin Han, Jinxiu Liang, Imari Sato, and Boxin Shi. Learning Event Guided High Dynamic Range Video Reconstruction. In *Proc. of Computer Vision and Pattern Recognition*, 2023.

[39] Zhichao Yin, Trevor Darrell, and Fisher Yu. Hierarchical discrete distribution decomposition for match density estimation. In *Proc. of Computer Vision and Pattern Recognition*, 2019.

[40] Dehao Zhang, Qiankun Ding, Peiqi Duan, Chu Zhou, and Boxin Shi. Data association between event streams and intensity frames under diverse baselines. In *Proc. of European Conference on Computer Vision*, 2022.

[41] Haoliang Zhao, Huizhou Zhou, Yongjun Zhang, Jie Chen, Yitong Yang, and Yong Zhao. High-frequency stereo matching network. In *Proc. of Computer Vision and Pattern Recognition*, 2023.

[42] Changyin Zhou and Shree K. Nayar. Computational cameras: Convergence of optics and processing. *IEEE Transactions on Image Processing*, 20(12):3322–3340, 2011.

[43] Yi Zhou, Guillermo Gallego, Henri Rebecq, Laurent Kneip, Hongdong Li, and Davide Scaramuzza. Semi-dense 3D reconstruction with a stereo event camera. In *Proc. of European Conference on Computer Vision*, 2018.

[44] Alex Zihao Zhu, Yibo Chen, and Kostas Daniilidis. Realtime time synchronized event-based stereo. In *Proc. of European Conference on Computer Vision*, 2018.

[45] Alex Zihao Zhu, Dinesh Thakur, Tolga Ozaslan, Bernd Pfrommer, Vijay Kumar, and Kostas Daniilidis. The multi vehicle stereo event camera dataset: An event camera dataset for 3D perception. *IEEE Robotics and Automation Letters*, 3(3):2032–2039, 2018.

[46] Yi-Fan Zuo, Li Cui, Xin Peng, Yanyu Xu, Shenghua Gao, Xia Wang, and Laurent Kneip. Accurate depth estimation from a hybrid event-rgb stereo setup. In *Proc. of International Conference on Intelligent Robots and Systems*, 2021.

# A  Appendix

This appendix provides additional implementation details and extended experimental results for the ZEST framework introduced in the main paper. We aim to facilitate the reproducibility and offer a more comprehensive analysis of the performance and robustness.

## A.1  Implementation Details

**Intermediate representation for stereo matching.**  Considering the physical formulation from frames to events, we design a representation that better captures the common information between the two modalities while suppressing their differences. Specifically, the proposed intermediate representation is designed to have the following properties: 1) It should be based on relative changes in intensity, which is the primary information captured by event cameras. 2) It should incorporate temporal information from the frames to match the temporal aggregation of events. 3) It should be robust to the different dynamic ranges and noise levels of event and frame data. By designing an intermediate representation with these properties, we aim to provide a more effective visual prompt for the off-the-shelf stereo matching models to adapt to the asymmetric characteristics of event and frame data. This can lead to an improved stereo matching performance in the event-intensity asymmetric setting compared to directly using the existing representations.

Now we provide the discrete form of the explicit representation defined in Eq. (7), which is used in practice since events with continuous time cannot be obtained. For convenience, we define the event map $E(t)$ as the integral of events between time $\tau$ and $\tau + \Delta\tau$ as $E_\tau(t)$ to represent the proportional change in intensity, which is equivalent to the sum of the polarity $\sigma_k$ of the $N_\tau$ events $e_k = (t_k, \boldsymbol{p}, \sigma_k)$ at position $\boldsymbol{p}$ in discrete form:

$$E_\tau(t) = \int_\tau^{\tau+\Delta\tau} e(t)dt = \sum_{t_k \in [\tau, \tau+\Delta\tau]} \sigma_k. \tag{16}$$

Suppose that the duration of the exposure time $2T$ is discretized into $N^{\mathrm{exp}}$ temporal bins with a predefined unit duration $\Delta\tau$. By ignoring the logarithm effects of events, the temporal difference $\Delta L_i(t)$ between two consecutive frames $L_{\tau_i}, L_{\tau_{i+1}}$ can be expressed into a reweighted sum form of brightness increment $E(t)$ as

$$\Delta\widehat{L}_i(t) = c\left( \sum_{\tau=\tau_i+T}^{\tau_{i+1}-T} N^{\mathrm{exp}} E_\tau + \sum_{\tau=\tau_{i+1}-T}^{\tau_{i+1}+T} \lfloor \frac{\tau_{i+1}+T-\tau}{\Delta\tau} \rfloor E_\tau - \sum_{\tau=\tau_i-T}^{\tau_i+T} \lfloor \frac{\tau-\tau_i+T}{\Delta\tau} \rfloor E_\tau \right),$$
$$\tag{17}$$

where $\lfloor \cdot \rfloor$ denotes the round down operation. Note that, compared to the commonly used event-based double integral model [24] that uses trilateral weights to reweigh the event bin, the weights used in the proposed method are trapezoidal, as shown on the left of Figure 2, where the events during the readout time between frames are weighted equally according to the physical formulations. This new formulation is especially useful when neither the exposure phase nor the readout phase is negligible towards each other. In summary, we use the temporal difference map $\Delta L(t)$ defined by consecutive frames in Eq. (4) and its approximation version defined from the temporal integral of events in Eq. (17) as explicit intermediate representations, respectively.

## A.2  More Qualitative Results

We provide additional qualitative comparisons of the disparity estimation results obtained by ZEST and state-of-the-art methods on the DSEC [13] dataset. The qualitative results of the baseline methods are shown in Figure 8 and Figure 9. Qualitative results of our methods and the compared methods are shown in Figure 10 and Figure 11. More qualitative results of the representation alignment method are shown in Figure 12. More intermediate results of the disparity refining method are shown in Figure 13. More results on diverse real-world scenes of our method are shown in Figure 14. The qualitative results of the proposed method on the MVSEC [45] and M3ED [2] datasets are shown in Figures 15 and 16.

## A.3  Computational Efficiency

Table 5: Computational complexity breakdown per stage. Runtime (ms), GPU memory usage (MB), number of parameters (M), and equivalent FPS are reported.

| Stage | Memory | Params | Runtime | FPS |
|---|---|---|---|---|
| Representation | 0 | – | 39.06 | 25.59 |
| DS | 9224 | 21.47 | 8515.32 | 0.11 |
| CR | 2078 | 5.43 | 243.55 | 4.11 |
| DA | 3640 | 335.32 | 79.99 | 12.5 |
| MiDaS | 3344 | 344.05 | 31.14 | 32.10 |
| Refinement | 1736 | – | 306.82 | 3.25 |

Table 6: Computational complexity analysis across methods. 3PE performance, runtime (ms), GPU memory usage (MB), number of parameters (M), and equivalent FPS are reported.

| Method | 3PE↓ | Memory | Params | Runtime | FPS |
|---|---|---|---|---|---|
| SHEF [30] | 54.37 | 0 | – | 28944.85 | 0.03 |
| HSM [17] | 33.08 | 766 | – | 224.85 | 4.44 |
| DAEI [40] | 86.96 | 3238 | 11.25 | 75.15 | 13.3 |
| Ours-DS-DA | 15.05 | 14600 | 356.79 | 8902.13 | 0.11 |
| Ours-DS-MiDaS | 14.91 | 14304 | 365.52 | 8853.27 | 0.11 |
| Ours-CR-DA | 9.84 | 7454 | 340.75 | 630.36 | 1.58 |
| Ours-CR-MiDaS | 29.26 | 7158 | 349.48 | 581.51 | 1.71 |

Table 7: Disparity refinement module computational cost across different iterations. EPE and 3PE performance, runtime (ms), and equivalent FPS are reported.

| Iterations | EPE↓ | 3PE↓ | Runtime | FPS |
|---|---|---|---|---|
| 0 | 1.487 | 7.785 | 4.20 | 238.06 |
| 50 | 1.488 | 8.028 | 42.39 | 23.59 |
| 100 | 1.451 | 7.457 | 70.92 | 14.10 |
| 200 | 1.430 | 7.270 | 127.75 | 7.82 |
| 300 | 1.420 | 7.234 | 188.14 | 5.31 |
| 400 | 1.413 | 7.227 | 247.63 | 4.03 |
| 500 (Ours) | 1.409 | 7.230 | 306.82 | 3.25 |

Table 8: Computational cost comparison at different input resolutions, reporting runtime (ms) and GPU memory usage (MB).

| Input | CRES | | DA | | Refinement | |
|---|---|---|---|---|---|---|
| | Runtime | Memory | Runtime | Memory | Runtime | Memory |
| 240×320 (1×) | 156.59 | 2064 | 81.27 | 3640 | 300.96 | 1688 |
| 480×640 (4×) | 243.55 | 2078 | 80.00 | 3640 | 306.82 | 1736 |
| 720×960 (9×) | 624.11 | 2738 | 80.26 | 3640 | 311.70 | 1808 |

We evaluated the computational efficiency of our method on an Intel i7-13700K CPU and a single NVIDIA RTX 4090 GPU, using a $480 \times 640$ input resolution. Unless noted otherwise, performance metrics were derived from the interlaken_00_c sequence of the DSEC dataset. Table 5 provides a breakdown of the computational cost per algorithm stage. The Ours-CR-DA variant averages 630.36 ms per frame, consuming 7454 MB of GPU memory. The disparity refinement module is the most computationally intensive, accounting for 48.6% of total runtime. In the Ours-DS-DA variant, the DS model bears most of the computational load, while the DA and refinement modules add minimal additional overhead. Table 6 compares the total computational cost of our method to existing methods.

Profiling results in Table 7 show the disparity refinement module requires approximately 306.82 ms, 48.6% of the Ours-CR-DA variant's total inference time (630.36 ms). This overhead can be reduced without significantly impacting performance by limiting the number of iterations.

We also evaluated scalability across input resolutions. Table 8 shows that GPU memory usage and runtime increase marginally with larger resolutions, mainly due to the DA model's fixed inference resolution, which stabilizes memory requirements.

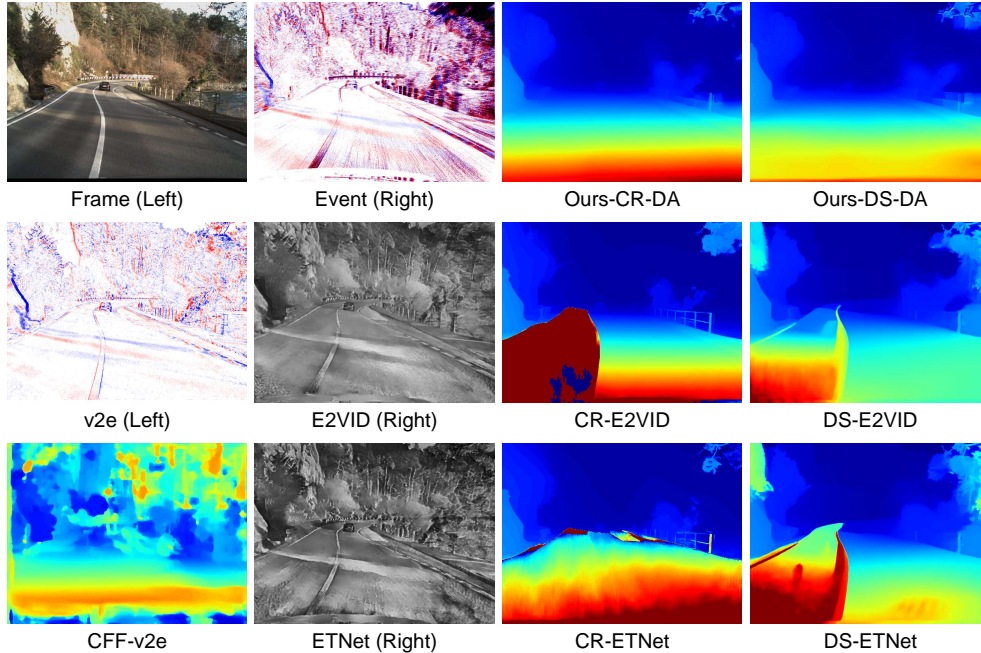

| Frame (Left) | Event (Right) | Ours-CR-DA | Ours-DS-DA |
| v2e (Left) | E2VID (Right) | CR-E2VID | DS-E2VID |
| CFF-v2e | ETNet (Right) | CR-ETNet | DS-ETNet |

Figure 8: Qualitative comparison of our method and baselines. Our methods demonstrate better robustness.

## A.4 Limitations

Despite the impressive performance of ZEST in event-intensity asymmetric stereo matching, there remain several limitations that warrant further investigation.

One challenge is handling noisy or sparse events, which can reduce the accuracy of visual prompts and stereo matching. In cases of significant noise or low event density, the disparity refinement module may struggle to compensate, resulting in suboptimal depth estimation. Representative failure cases are illustrated in Figure 17. Row 1 shows how noisy events increase the visual discrepancy between views, leading to stereo model errors partially mitigated by monocular DA predictions but ultimately producing suboptimal results. Row 2 depicts the challenges of sparse events, where limited event information hampers stereo matching, and the refinement module struggled to compensate.

Additionally, The representation alignment module employed in the current framework relies on a fixed transformation, which may not fully capture the intricacies of the modality gap between events and frames. Future research could explore more expressive modality alignment techniques, such as learning-based approaches or domain adaptation methods, to improve the robustness and generalization capabilities of the framework.

Furthermore, the use of off-the-shelf image-domain models adds considerable computational load due to their large parameter counts. Nevertheless, ZEST's modular design allows for lightweight alternatives to be substituted in place of the stereo and monocular depth estimation models. This flexibility provides options for resource-limited deployments, though with some trade-offs in accuracy.

## A.5 Broader Impacts

The proposed ZEST framework has the potential to significantly advance the field of event-intensity asymmetric stereo matching and enable a wide range of applications in various domains. In autonomous driving, the improved disparity estimation provided by ZEST can contribute to better obstacle detection, 3D object location, and scene understanding, ultimately improving the safety and reliability of self-driving vehicles. In robotics, the enhanced depth perception enabled by our method can facilitate more precise object manipulation, navigation, and mapping tasks, particularly in dynamic environments where conventional frame-based cameras may struggle. Furthermore, the zero-shot learning approach of ZEST lowers the entry barrier for researchers and practitioners to explore the benefits of event-intensity asymmetric stereo matching in their specific fields, as it eliminates the need for large-scale labeled training data.

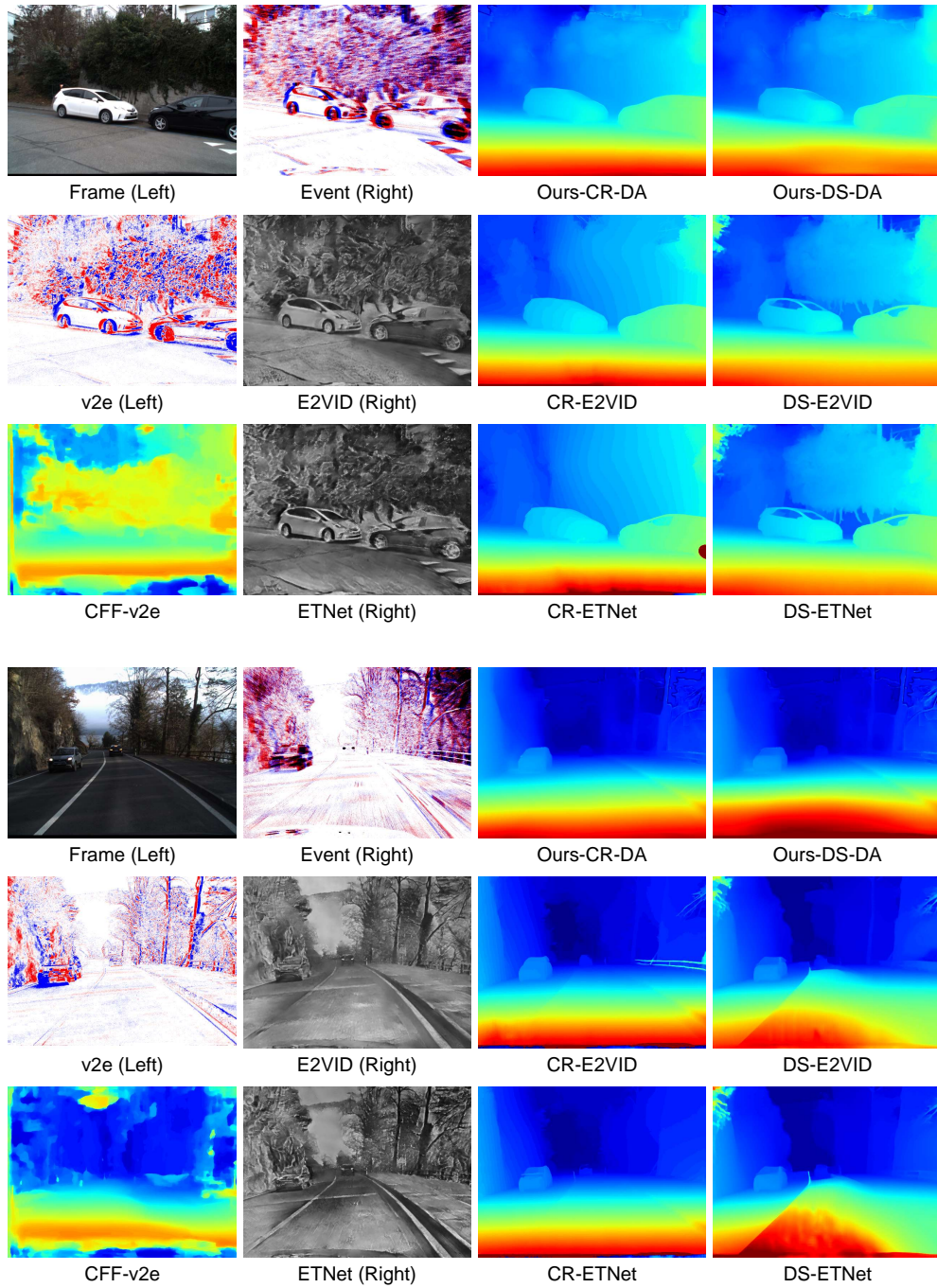

Figure 9: Qualitative comparison of our method and baselines. Our methods demonstrate better robustness.

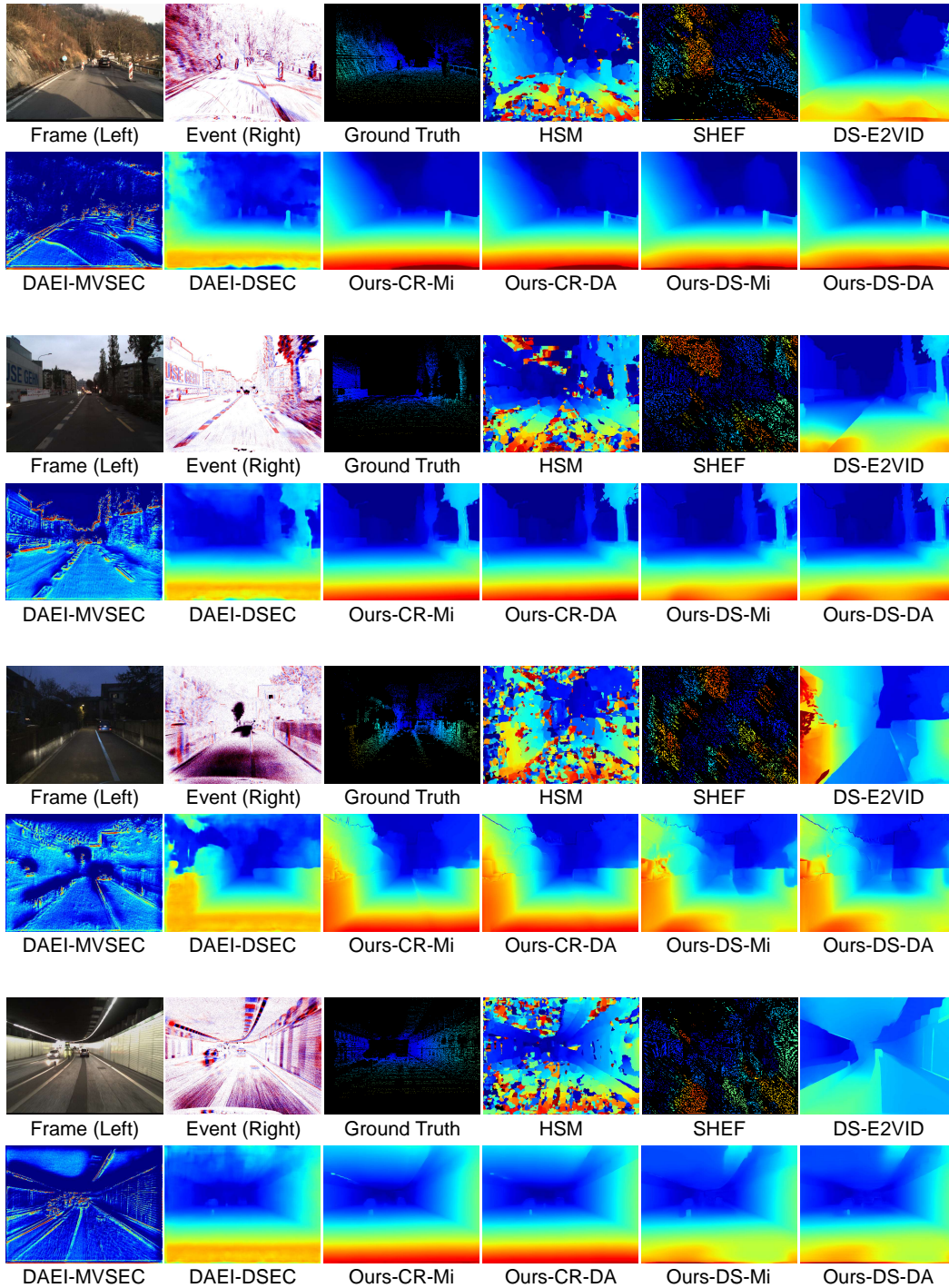

Figure 10: Qualitative comparison of our method and other methods.

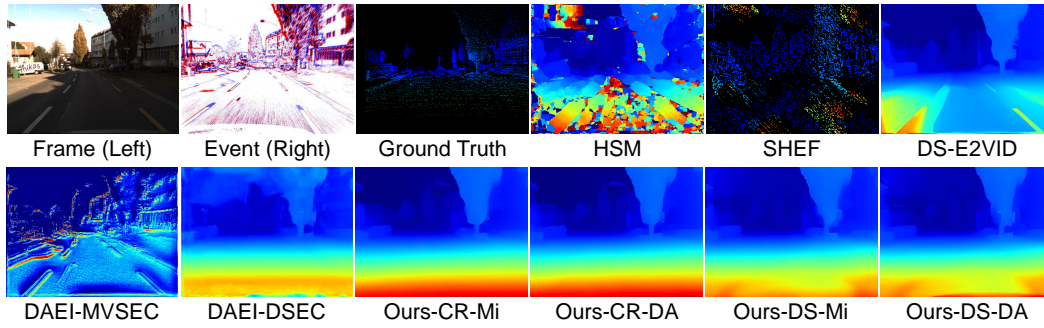

Figure 11: Qualitative comparison of our method and other methods.

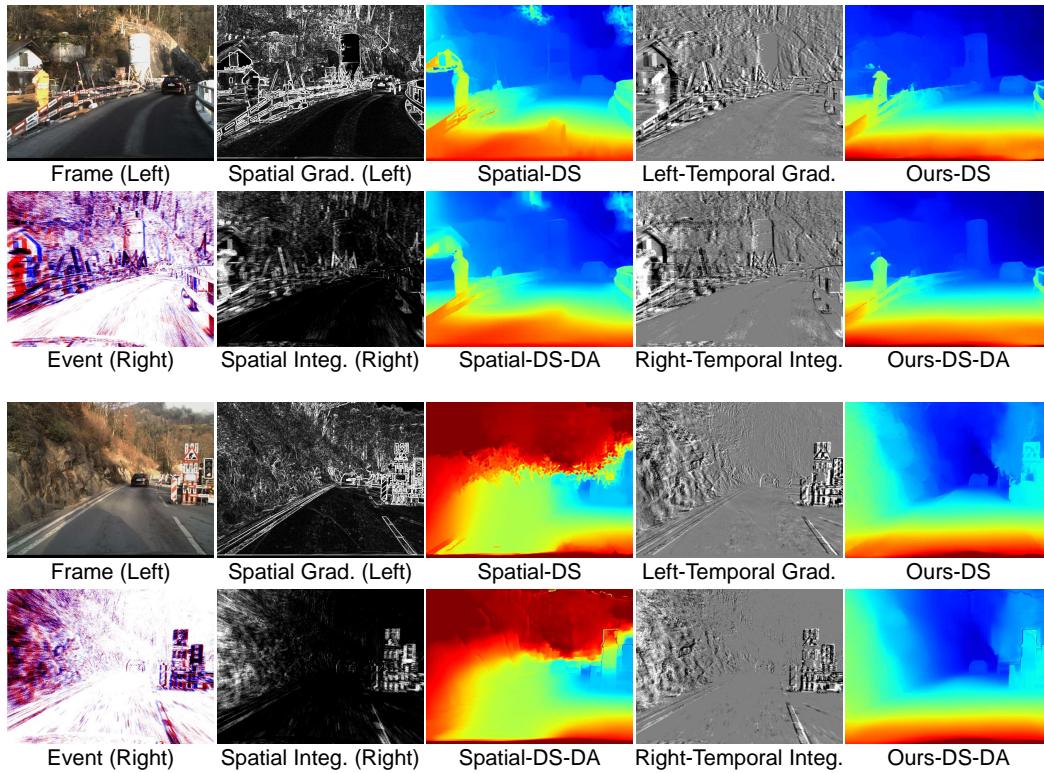

Figure 12: Qualitative comparison of our method and other representations.

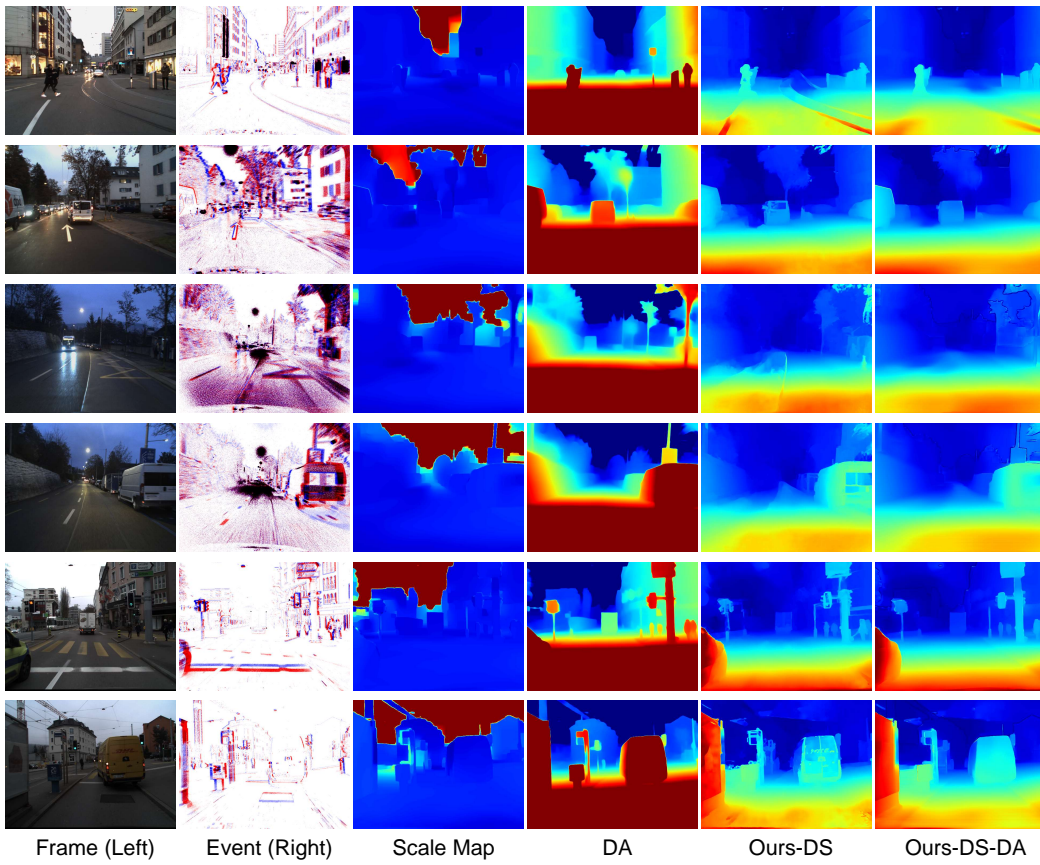

| Frame (Left) | Event (Right) | Scale Map | DA | Ours-DS | Ours-DS-DA |

Figure 13: Intermediate results of disparity refining.

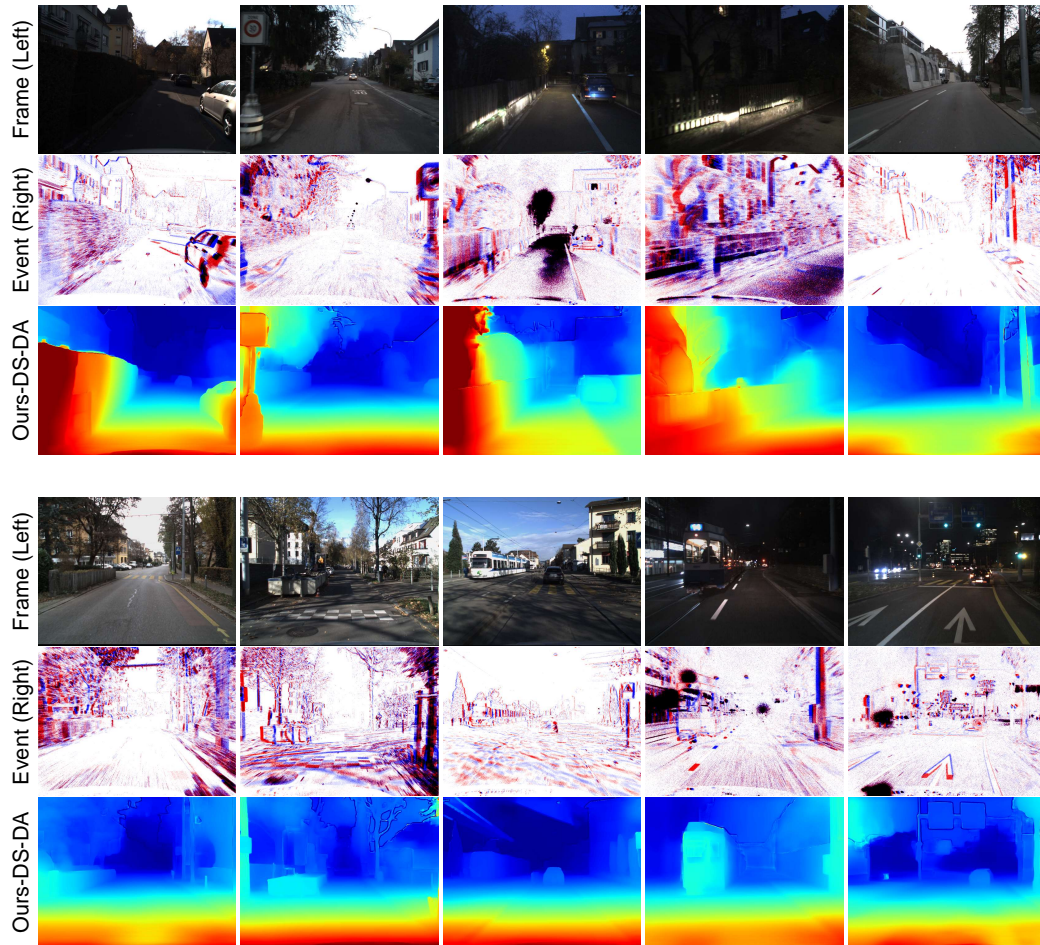

Figure 14: Results of our method in diverse scenarios.

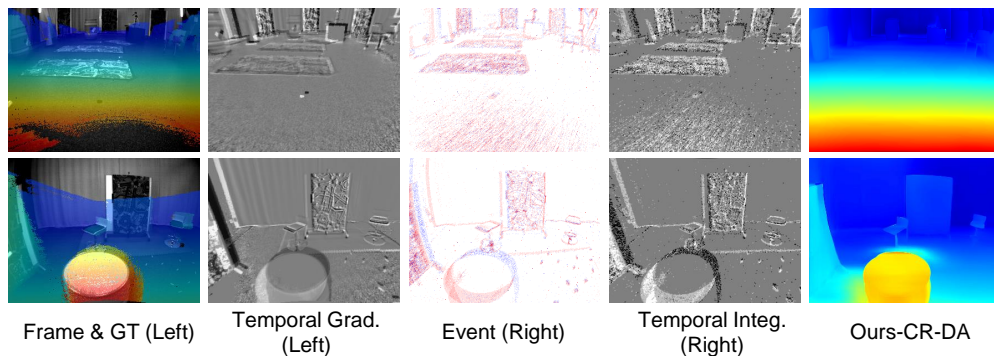

| Frame & GT (Left) | Temporal Grad. (Left) | Event (Right) | Temporal Integ. (Right) | Ours-CR-DA |

Figure 15: Comparison of disparity estimation results for real data from the MVSEC [45] dataset.

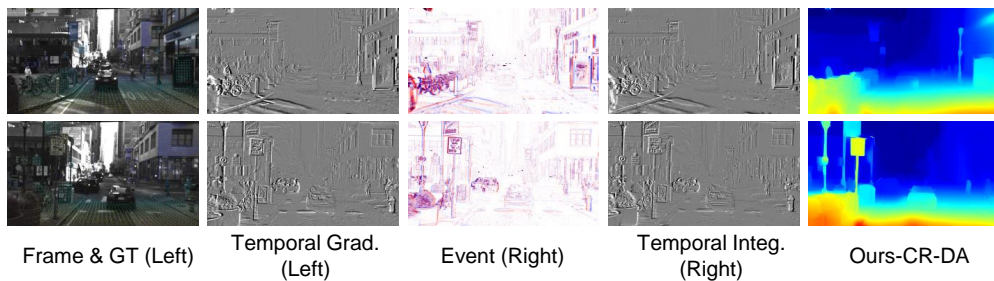

| Frame & GT (Left) | Temporal Grad. (Left) | Event (Right) | Temporal Integ. (Right) | Ours-CR-DA |

Figure 16: Comparison of disparity estimation results for real data from the M3ED [2] dataset.

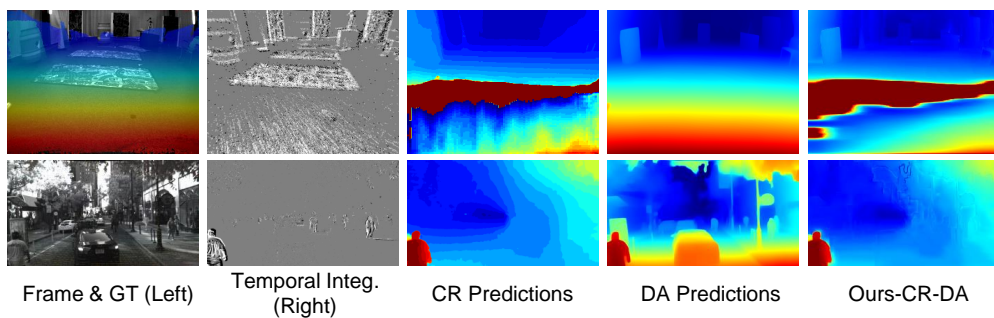

| Frame & GT (Left) | Temporal Integ. (Right) | CR Predictions | DA Predictions | Ours-CR-DA |

Figure 17: Examples of failure cases for the proposed method.

